# Monaural Speech Separation

**Guoning Hu**
Biophysics Program
The Ohio State University
Columbus, OH 43210
*hu.117@osu.edu*

**DeLiang Wang**
Department of Computer and Information
Science & Center of Cognitive Science
The Ohio State University, Columbus, OH 43210
*dwang@cis.ohio-state.edu*

## Abstract

Monaural speech separation has been studied in previous systems that incorporate auditory scene analysis principles. A major problem for these systems is their inability to deal with speech in the high-frequency range. Psychoacoustic evidence suggests that different perceptual mechanisms are involved in handling resolved and unresolved harmonics. Motivated by this, we propose a model for monaural separation that deals with low-frequency and high-frequency signals differently. For resolved harmonics, our model generates segments based on temporal continuity and cross-channel correlation, and groups them according to periodicity. For unresolved harmonics, the model generates segments based on amplitude modulation (AM) in addition to temporal continuity and groups them according to AM repetition rates derived from sinusoidal modeling. Underlying the separation process is a pitch contour obtained according to psychoacoustic constraints. Our model is systematically evaluated, and it yields substantially better performance than previous systems, especially in the high-frequency range.

## 1  Introduction

In a natural environment, speech usually occurs simultaneously with acoustic interference. An effective system for attenuating acoustic interference would greatly facilitate many applications, including automatic speech recognition (ASR) and speaker identification. Blind source separation using independent component analysis [10] or sensor arrays for spatial filtering require multiple sensors. In many situations, such as telecommunication and audio retrieval, a monaural (one microphone) solution is required, in which intrinsic properties of speech or interference must be considered. Various algorithms have been proposed for monaural speech enhancement [14]. These methods assume certain properties of interference and have difficulty in dealing with general acoustic interference. Monaural separation has also been studied using phase-based decomposition [3] and statistical learning [17], but with only limited evaluation.

While speech enhancement remains a challenge, the auditory system shows a remarkable capacity for monaural speech separation. According to Bregman [1], the auditory system separates the acoustic signal into streams, corresponding to different sources, based on auditory scene analysis (ASA) principles. Research in ASA has inspired considerable work to build computational auditory scene analysis (CASA)

systems for sound separation [19] [4] [7] [18]. Such systems generally approach speech separation in two main stages: segmentation (analysis) and grouping (synthesis). In segmentation, the acoustic input is decomposed into sensory segments, each of which is likely to originate from a single source. In grouping, those segments that likely come from the same source are grouped together, based mostly on periodicity. In a recent CASA model by Wang and Brown [18], segments are formed on the basis of similarity between adjacent filter responses (cross-channel correlation) and temporal continuity, while grouping among segments is performed according to the global pitch extracted within each time frame. In most situations, the model is able to remove intrusions and recover low-frequency (below 1 kHz) energy of target speech. However, this model cannot handle high-frequency (above 1 kHz) signals well, and it loses much of target speech in the high-frequency range. In fact, the inability to deal with speech in the high-frequency range is a common problem for CASA systems.

We study monaural speech separation with particular emphasis on the high-frequency problem in CASA. For voiced speech, we note that the auditory system can resolve the first few harmonics in the low-frequency range [16]. It has been suggested that different perceptual mechanisms are used to handle resolved and unresolved harmonics [2]. Consequently, our model employs different methods to segregate resolved and unresolved harmonics of target speech. More specifically, our model generates segments for resolved harmonics based on temporal continuity and cross-channel correlation, and these segments are grouped according to common periodicity. For unresolved harmonics, it is well known that the corresponding filter responses are strongly amplitude-modulated and the response envelopes fluctuate at the fundamental frequency (F0) of target speech [8]. Therefore, our model generates segments for unresolved harmonics based on common AM in addition to temporal continuity. The segments are grouped according to AM repetition rates. We calculate AM repetition rates via sinusoidal modeling, which is guided by target pitch estimated according to characteristics of natural speech.

Section 2 describes the overall system. In section 3, systematic results and a comparison with the Wang-Brown system are given. Section 4 concludes the paper.

## 2 Model description

Our model is a multistage system, as shown in Fig. 1. Description for each stage is given below.

### 2.1 Initial processing

First, an acoustic input is analyzed by a standard cochlear filtering model with a bank of 128 gammatone filters [15] and subsequent hair cell transduction [12]. This peripheral processing is done in time frames of 20 ms long with 10 ms overlap between consecutive frames. As a result, the input signal is decomposed into a group of time-frequency (T-F) units. Each T-F unit contains the response from a certain channel at a certain frame. The envelope of the response is obtained by a lowpass filter with

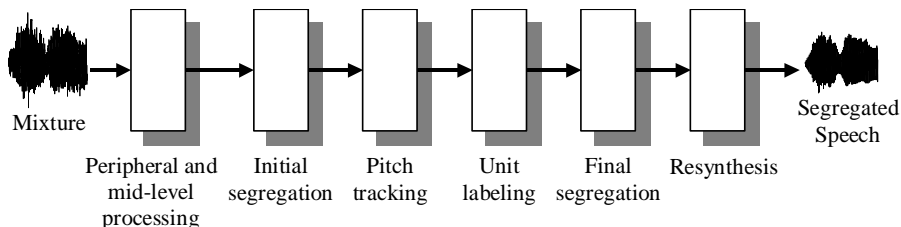

**Figure 1**. Schematic diagram of the proposed multistage system.

passband [0, 1 kHz] and a Kaiser window of 18.25 ms.

Mid-level processing is performed by computing a correlogram (autocorrelation function) of the individual responses and their envelopes. These autocorrelation functions reveal response periodicities as well as AM repetition rates. The global pitch is obtained from the summary correlogram. For clean speech, the autocorrelations generally have peaks consistent with the pitch and their summation shows a dominant peak corresponding to the pitch period. With acoustic interference, a global pitch may not be an accurate description of the target pitch, but it is reasonably close.

Because a harmonic extends for a period of time and its frequency changes smoothly, target speech likely activates contiguous T-F units. This is an instance of the temporal continuity principle. In addition, since the passbands of adjacent channels overlap, a resolved harmonic usually activates adjacent channels, which leads to high cross-channel correlations. Hence, in initial segregation, the model first forms segments by merging T-F units based on temporal continuity and cross-channel correlation. Then the segments are grouped into a foreground stream and a background stream by comparing the periodicities of unit responses with global pitch. A similar process is described in [18].

Fig. 2(a) and Fig. 2(b) illustrate the segments and the foreground stream. The input is a mixture of a voiced utterance and a cocktail party noise (see Sect. 3). Since the intrusion is not strongly structured, most segments correspond to target speech. In addition, most segments are in the low-frequency range. The initial foreground stream successfully groups most of the major segments.

## 2.2 Pitch tracking

In the presence of acoustic interference, the global pitch estimated in mid-level processing is generally not an accurate description of target pitch. To obtain accurate pitch information, target pitch is first estimated from the foreground stream. At each frame, the autocorrelation functions of T-F units in the foreground stream are summated. The pitch period is the lag corresponding to the maximum of the summation in the plausible pitch range: [2 ms, 12.5 ms]. Then we employ the following two constraints to check its reliability. First, an accurate pitch period at a frame should be consistent with the periodicity of the T-F units at this frame in the foreground stream. At frame $j$, let $\tau(j)$ represent the estimated pitch period, and $A(i,j,\tau)$ the autocorrelation function of $u_{ij}$, the unit in channel $i$. $u_{ij}$ agrees with $\tau(j)$ if

$$A(i, j, \tau(j)) / A(i, j, \tau_m) > \theta_d \tag{1}$$

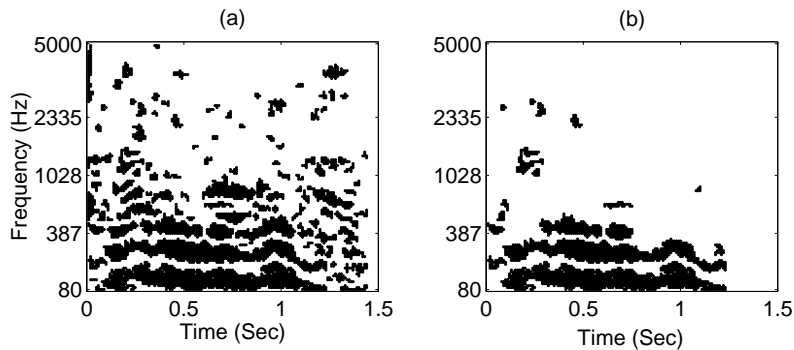

**Figure 2**. Results of initial segregation for a speech and cocktail-party mixture. (a) Segments formed. Each segment corresponds to a contiguous black region. (b) Foreground stream.

Here, $\theta_d=0.95$, the same threshold used in [18], and $\tau_m$ is the lag corresponding to the maximum of $A(i,j,\tau)$ within [2 ms, 12.5 ms]. $\tau(j)$ is considered reliable if more than half of the units in the foreground stream at frame $j$ agree with it. Second, pitch periods in natural speech vary smoothly in time [11]. We stipulate the difference between reliable pitch periods at consecutive frames be smaller than 20% of the pitch period, justified from pitch statistics. Unreliable pitch periods are replaced by new values extrapolated from reliable pitch points using temporal continuity. As an example, suppose at two consecutive frames $j$ and $j+1$ that $\tau(j)$ is reliable while $\tau(j+1)$ is not. All the channels corresponding to the T-F units agreeing with $\tau(j)$ are selected. $\tau(j+1)$ is then obtained from the summation of the autocorrelations for the units at frame $j+1$ in those selected channels. Then the re-estimated pitch is further verified with the second constraint. For more details, see [9].

Fig. 3 illustrates the estimated pitch periods from the speech and cocktail-party mixture, which match the pitch periods obtained from clean speech very well.

## 2.3  Unit labeling

With estimated pitch periods, (1) provides a criterion to label T-F units according to whether target speech dominates the unit responses or not. This criterion compares an estimated pitch period with the periodicity of the unit response. It is referred as the periodicity criterion. It works well for resolved harmonics, and is used to label the units of the segments generated in initial segregation.

However, the periodicity criterion is not suitable for units responding to multiple harmonics because unit responses are amplitude-modulated. As shown in Fig. 4, for a filter response that is strongly amplitude-modulated (Fig. 4(a)), the target pitch corresponds to a local maximum, indicated by the vertical line, in the autocorrelation instead of the global maximum (Fig. 4(b)). Observe that for a filter responding to multiple harmonics of a harmonic source, the response envelope fluctuates at the rate of F0 [8]. Hence, we propose a new criterion for labeling the T-F units corresponding to unresolved harmonics by comparing AM repetition rates with estimated pitch. This criterion is referred as the AM criterion.

To obtain an AM repetition rate, the entire response of a gammatone filter is half-wave rectified and then band-pass filtered to remove the DC component and other possible

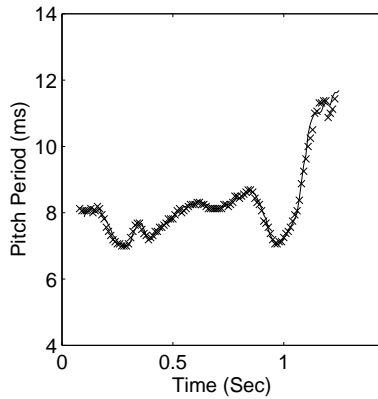

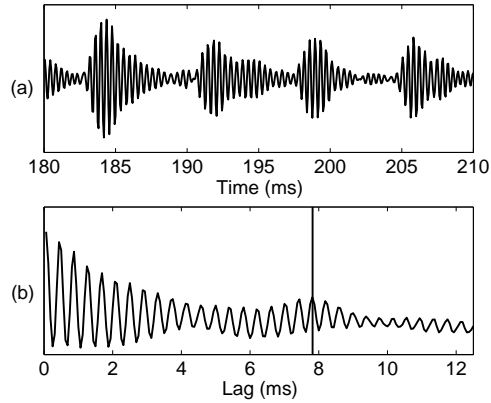

**Figure 3**. Estimated target pitch for the speech and cocktail-party mixture, marked by "x". The solid line indicates the pitch contour obtained from clean speech.

**Figure 4**. AM effects. (a) Response of a filter with center frequency 2.6 *k*Hz. (b) Corresponding autocorrelation. The vertical line marks the position corresponding to the pitch period of target speech.

harmonics except for the F0 component. The rectified and filtered signal is then normalized by its envelope to remove the intensity fluctuations of the original signal, where the envelope is obtained via the Hilbert Transform. Because the pitch of natural speech does not change noticeably within a single frame, we model the corresponding normalized signal within a T-F unit by a single sinusoid to obtain the AM repetition rate. Specifically,

$$f_{ij}, \phi_{ij} = \arg\min_{f,\phi} \sum_{k=1}^{M} [\hat{r}(i, jT - k) - \sin(2\pi k f / f_S + \phi)]^2, \text{ for } f \in [80 \text{ Hz, } 500 \text{ Hz}], \quad (2)$$

where a square error measure is used. $\hat{r}(i,t)$ is the normalized filter response, $f_S$ is the sampling frequency, $M$ spans a frame, and $T = 10$ ms is the progressing period from one frame to the next. In the above equation, $f_{ij}$ gives the AM repetition rate for unit $u_{ij}$. Note that in the discrete case, a single sinusoid with a sufficiently high frequency can always match these samples perfectly. However, we are interested in finding a frequency within the plausible pitch range. Hence, the solution does not reduce to a degenerate case. With appropriately chosen initial values, this optimization problem can be solved effectively using iterative gradient descent (see [9]).

The AM criterion is used to label T-F units that do not belong to any segments generated in initial segregation; such segments, as discussed earlier, tend to miss unresolved harmonics. Specifically, unit $u_{ij}$ is labeled as target speech if the final square error is less than half of the total energy of the corresponding signal and the AM repetition rate is close to the estimated target pitch:

$$| f_{ij}\tau(j) - 1 | < \theta_f. \quad (3)$$

Psychoacoustic evidence suggests that to separate sounds with overlapping spectra requires 6-12% difference in F0 [6]. Accordingly, we choose $\theta_f$ to be 0.12.

## 2.4 Final segregation and resynthesis

For adjacent channels responding to unresolved harmonics, although their responses may be quite different, they exhibit similar AM patterns and their response envelopes are highly correlated. Therefore, for T-F units labeled as target speech, segments are generated based on cross-channel envelope correlation in addition to temporal continuity.

The spectra of target speech and intrusion often overlap and, as a result, some segments generated in initial segregation contain both units where target speech dominates and those where intrusion dominates. Given unit labels generated in the last stage, we further divide the segments in the foreground stream, $S_F$, so that all the units in a segment have the same label. Then the streams are adjusted as follows. First, since segments for speech usually are at least 50 ms long, segments with the target label are retained in $S_F$ only if they are no shorter than 50 ms. Second, segments with the intrusion label are added to the background stream, $S_B$, if they are no shorter than 50 ms. The remaining segments are removed from $S_F$, becoming undecided.

Finally, other units are grouped into the two streams by temporal and spectral continuity. First, $S_B$ expands iteratively to include undecided segments in its neighborhood. Then, all the remaining undecided segments are added back to $S_F$. For individual units that do not belong to either stream, they are grouped into $S_F$ iteratively if the units are labeled as target speech as well as in the neighborhood of $S_F$. The resulting $S_F$ is the final segregated stream of target speech.

Fig. 5(a) shows the new segments generated in this process for the speech and cocktail-party mixture. Fig. 5(b) illustrates the segregated stream from the same mixture. Fig. 5(c) shows all the units where target speech is stronger than intrusion. The foreground

stream generated by our algorithm contains most of the units where target speech is stronger. In addition, only a small number of units where intrusion is stronger are incorrectly grouped into it.

A speech waveform is resynthesized from the final foreground stream. Here, the foreground stream works as a binary mask. It is used to retain the acoustic energy from the mixture that corresponds to 1's and reject the mixture energy corresponding to 0's. For more details, see [19].

## 3   Evaluation and comparison

Our model is evaluated with a corpus of 100 mixtures composed of 10 voiced utterances mixed with 10 intrusions collected by Cooke [4]. The intrusions have a considerable variety. Specifically, they are: N0 - 1 $k$Hz pure tone, N1 - white noise, N2 - noise bursts, N3 - "cocktail party" noise, N4 - rock music, N5 - siren, N6 - trill telephone, N7 - female speech, N8 - male speech, and N9 - female speech.

Given our decomposition of an input signal into T-F units, we suggest the use of an ideal binary mask as the ground truth for target speech. The ideal binary mask is constructed as follows: a T-F unit is assigned one if the target energy in the corresponding unit is greater than the intrusion energy and zero otherwise. Theoretically speaking, an ideal binary mask gives a performance ceiling for all binary masks. Figure 5(c) illustrates the ideal mask for the speech and cocktail-party mixture. Ideal masks also suit well the situations where more than one target need to be segregated or the target changes dynamically. The use of ideal masks is supported by the auditory masking phenomenon: within a critical band, a weaker signal is masked by a stronger one [13]. In addition, an ideal mask gives excellent resynthesis for a variety of sounds and is similar to a prior mask used in a recent ASR study that yields excellent recognition performance [5].

The speech waveform resynthesized from the final foreground stream is used for evaluation, and it is denoted by $S(t)$. The speech waveform resynthesized from the ideal binary mask is denoted by $I(t)$. Furthermore, let $e_1(t)$ denote the signal present in $I(t)$ but missing from $S(t)$, and $e_2(t)$ the signal present in $S(t)$ but missing from $I(t)$. Then, the relative energy loss, $R_{EL}$, and the relative noise residue, $R_{NR}$, are calculated as follows:

$$R_{EL} = \sum_t e_1^2(t) \bigg/ \sum_t I^2(t) , \tag{4a}$$

$$R_{NR} = \sum_t e_2^2(t) \bigg/ \sum_t S^2(t) . \tag{4b}$$

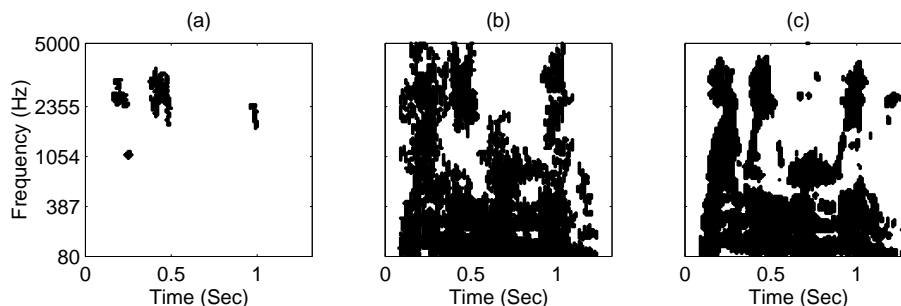

**Figure 5**. Results of final segregation for the speech and cocktail-party mixture. (a) New segments formed in the final segregation. (b) Final foreground stream. (c) Units where target speech is stronger than the intrusion.

**Table 1**: $R_{EL}$ and $R_{NR}$

| Intrusion | Proposed model | | Wang-Brown model | |
|---|---|---|---|---|
| | $R_{EL}$ (%) | $R_{NR}$ (%) | $R_{EL}$ (%) | $R_{NR}$ (%) |
| N0 | 2.12 | 0.02 | 6.99 | 0 |
| N1 | 4.66 | 3.55 | 28.96 | 1.61 |
| N2 | 1.38 | 1.30 | 5.77 | 0.71 |
| N3 | 3.83 | 2.72 | 21.92 | 1.92 |
| N4 | 4.00 | 2.27 | 10.22 | 1.41 |
| N5 | 2.83 | 0.10 | 7.47 | 0 |
| N6 | 1.61 | 0.30 | 5.99 | 0.48 |
| N7 | 3.21 | 2.18 | 8.61 | 4.23 |
| N8 | 1.82 | 1.48 | 7.27 | 0.48 |
| N9 | 8.57 | 19.33 | 15.81 | 33.03 |
| **Average** | 3.40 | 3.32 | 11.91 | 4.39 |

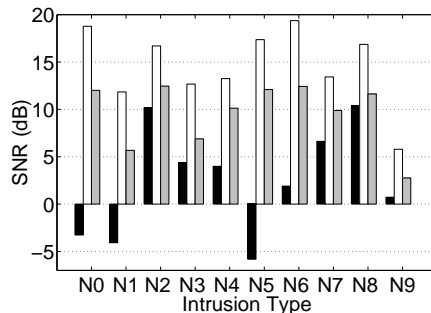

**Figure 6**. SNR results for segregated speech. White bars show the results from the proposed model, gray bars those from the Wang-Brown system, and black bars those of the mixtures.

The results from our model are shown in Table 1. Each value represents the average of one intrusion with 10 voiced utterances. A further average across all intrusions is also shown in the table. On average, our system retains 96.60% of target speech energy, and the relative residual noise is kept at 3.32%. As a comparison, Table 1 also shows the results from the Wang-Brown model [18], whose performance is representative of current CASA systems. As shown in the table, our model reduces $R_{EL}$ significantly. In addition, $R_{EL}$ and $R_{NR}$ are balanced in our system.

Finally, to compare waveforms directly we measure a form of signal-to-noise ratio (SNR) in decibels using the resynthesized signal from the ideal binary mask as ground truth:

$$SNR = 10\log_{10}[\sum_t I^2(t) \Big/ \sum_t (I(t) - S(t))^2]. \qquad (5)$$

The SNR for each intrusion averaged across 10 target utterances is shown in Fig. 6, together with the results from the Wang-Brown system and the SNR of the original mixtures. Our model achieves an average SNR gain of around 12 dB and 5 dB improvement over the Wang-Brown model.

## 4  Discussion

The main feature of our model lies in using different mechanisms to deal with resolved and unresolved harmonics. As a result, our model is able to recover target speech and reduce noise interference in the high-frequency range where harmonics of target speech are unresolved.

The proposed system considers the pitch contour of the target source only. However, it is possible to track the pitch contour of the intrusion if it has a harmonic structure. With two pitch contours, one could label a T-F unit more accurately by comparing whether its periodicity is more consistent with one or the other. Such a method is expected to lead to better performance for the two-speaker situation, e.g. N7 through N9. As indicated in Fig. 6, the performance gain of our system for such intrusions is relatively limited. Our model is limited to separation of voiced speech. In our view, unvoiced speech poses the biggest challenge for monaural speech separation. Other grouping cues, such as onset, offset, and timbre, have been demonstrated to be effective for human ASA [1], and may play a role in grouping unvoiced speech. In addition, one should consider the acoustic and phonetic characteristics of individual unvoiced consonants. We plan to investigate these issues in future study.

## Acknowledgments

We thank G. J. Brown and M. Wu for helpful comments. Preliminary versions of this work were presented in 2001 IEEE WASPAA and 2002 IEEE ICASSP. This research was supported in part by an NSF grant (IIS-0081058) and an AFOSR grant (F49620-01-1-0027).

## References

[1] A. S. Bregman, *Auditory scene analysis*, Cambridge MA: MIT Press, 1990.

[2] R. P. Carlyon and T. M. Shackleton, "Comparing the fundamental frequencies of resolved and unresolved harmonics: evidence for two pitch mechanisms?" *J. Acoust. Soc. Am.*, Vol. 95, pp. 3541-3554, 1994.

[3] G. Cauwenberghs, "Monaural separation of independent acoustical components," In *Proc. of IEEE Symp. Circuit & Systems*, 1999.

[4] M. Cooke, *Modeling auditory processing and organization*, Cambridge U.K.: Cambridge University Press, 1993.

[5] M. Cooke, P. Green, L. Josifovski, and A. Vizinho, "Robust automatic speech recognition with missing and unreliable acoustic data," *Speech Comm.*, Vol. 34, pp. 267-285, 2001.

[6] C. J. Darwin and R. P. Carlyon, "Auditory grouping," in *Hearing*, B. C. J. Moore, Ed., San Diego CA: Academic Press, 1995.

[7] D. P. W. Ellis, *Prediction-driven computational auditory scene analysis*, Ph.D. Dissertation, MIT Department of Electrical Engineering and Computer Science, 1996.

[8] H. Helmholtz, *On the sensations of tone*, Braunschweig: Vieweg & Son, 1863. (A. J. Ellis, English Trans., Dover, 1954.)

[9] G. Hu and D. L. Wang, "Monaural speech segregation based on pitch tracking and amplitude modulation," *Technical Report* TR6, Ohio State University Department of Computer and Information Science, 2002. (available at www.cis.ohio-state.edu/~hu)

[10] A. Hyvärinen, J. Karhunen, and E. Oja, *Independent component analysis*, New York: Wiley, 2001.

[11] W. J. M. Levelt, *Speaking: From intention to articulation*, Cambridge MA: MIT Press, 1989.

[12] R. Meddis, "Simulation of auditory-neural transduction: further studies," *J. Acoust. Soc. Am.*, Vol. 83, pp. 1056-1063, 1988.

[13] B. C. J. Moore, *An Introduction to the psychology of hearing*, 4th Ed., San Diego CA: Academic Press, 1997.

[14] D. O'Shaughnessy, *Speech communications: human and machine*, 2nd Ed., New York: IEEE Press, 2000.

[15] R. D. Patterson, I. Nimmo-Smith, J. Holdsworth, and P. Rice, "An efficient auditory filterbank based on the gammatone function," *APU Report* 2341, MRC, Applied Psychology Unit, Cambridge U.K., 1988.

[16] R. Plomp and A. M. Mimpen, "The ear as a frequency analyzer II," *J. Acoust. Soc. Am.*, Vol. 43, pp. 764-767, 1968.

[17] S. Roweis, "One microphone source separation," In *Advances in Neural Information Processing Systems* 13 (NIPS'00), 2001.

[18] D. L. Wang and G. J. Brown, "Separation of speech from interfering sounds based on oscillatory correlation," *IEEE Trans. Neural Networks*, Vol. 10, pp. 684-697, 1999.

[19] M. Weintraub, *A theory and computational model of auditory monaural sound separation*, Ph.D. Dissertation, Stanford University Department of Electrical Engineering, 1985.
